# Semi-Supervised MarginBoost

**F. d'Alché-Buc**
LIP6,UMR CNRS 7606,
Université P. et M. Curie
75252 Paris Cedex, France
*florence.dAlche@lip6.fr*

**Yves Grandvalet**
Heudiasyc, UMR CNRS 6599,
Université de Technologie de Compiègne,
BP 20.529, 60205 Compiègne cedex, France
*Yves.Grandvalet@hds.utc.fr*

**Christophe Ambroise**
Heudiasyc, UMR CNRS 6599,
Université de Technologie de Compiègne,
BP 20.529, 60205 Compiègne cedex, France
*Christophe Ambroise@hds.utc.fr*

## Abstract

In many discrimination problems a large amount of data is available but only a few of them are labeled. This provides a strong motivation to improve or develop methods for semi-supervised learning. In this paper, boosting is generalized to this task within the optimization framework of MarginBoost . We extend the margin definition to unlabeled data and develop the gradient descent algorithm that corresponds to the resulting margin cost function. This meta-learning scheme can be applied to any base classifier able to benefit from unlabeled data. We propose here to apply it to mixture models trained with an Expectation-Maximization algorithm. Promising results are presented on benchmarks with different rates of labeled data.

## 1 Introduction

In semi-supervised classification tasks, a concept is to be learnt using both labeled and unlabeled examples. Such problems arise frequently in data-mining where the cost of the labeling process can be prohibitive because it requires human help as in video-indexing, text-categorization [12] and medical diagnosis. While some works proposed different methods [16] to learn mixture models [12], [1], SVM [3], co-trained machines [5] to solve this task, no extension has been developed so far for ensemble methods such as boosting [7, 6]. Boosting consists in building sequentially a linear combination of base classifiers that focus on the difficult examples. For AdaBoost and extensions such as MarginBoost [10], this stage-wise procedure corresponds to a gradient descent of a cost functional based on a decreasing function of the margin, in the space of linear combinations of base classifiers.

We propose to generalize boosting to semi-supervised learning within the framework of optimization. We extend the margin notion to unlabeled data, derive the corresponding criterion to be maximized, and propose the resulting algorithm called Semi-Supervised MarginBoost (SSMBoost). This new method enhances our previ-

ous work [9] based on a direct plug-in extension of AdaBoost in the sense that all the ingredients of the gradient algorithm such as the gradient direction and the stopping rule are defined from the expression of the new cost function. Moreover, while the algorithm has been tested using the mixtures of models [1], SSMBoost is designed to combine any base classifiers that deals with both labeled and unlabeled data. The paper begins with a brief presentation of MarginBoost (section 2). Then, in section 3, the SSMBoost algorithm is presented. Experimental results are discussed in section 5 and we conclude in section 6.

## 2 Boosting with MarginBoost

Boosting [7, 6, 15] aims at improving the performance of any weak "base classifier" by linear combination. We focus here on normalized ensemble classifiers $g_t \in Lin(\mathcal{H})$ whose normalized[1] coefficients are noted $\bar{\alpha}_\tau = \frac{\alpha_\tau}{|\alpha_\tau|}$ and each base classifier with outputs in $[-1, 1]$ is $h_\tau \in \mathcal{H}$:

$$g_t(\mathbf{x}) = \sum_{\tau=1}^{t} \bar{\alpha}_\tau h_\tau(\mathbf{x}) \qquad (1)$$

Different contributions [13, 14],[8], [10] have described boosting within an optimization scheme, considering that it carries out a gradient descent in the space of linear combinations of base functions. We have chosen the MarginBoost algorithm, a variant of a more general algorithm called AnyBoost [10], that generalizes AdaBoost and formally justifies the interpretation in terms of margin. If $S$ is the training sample $\{(\mathbf{x}_i, y_i), i = 1..l\}$, MarginBoost, described in Fig. 1, minimizes the cost functional $C$ defined for any scalar decreasing function $c$ of the margin $\rho$ :

$$C(g_t) = \sum_{i=1}^{l} c(\rho(g_t(\mathbf{x}_i), y_i))) \qquad \text{with } \rho(g_t(\mathbf{x}_i), y_i) = y_i g_t(\mathbf{x}_i) \qquad (2)$$

Instead of taking exactly $h_{t+1} = -\nabla C(g_t)$ which does not ensure that the resulting function $g_{t+1}$ belongs to $Lin(\mathcal{H})$, $h_{t+1}$ is chosen such as the inner product[2] $- < \nabla C(g_t), h_{t+1} >$ is maximal. The equivalent weighted cost function to be maximized can thus be expressed as :

$$J_t^L = \sum_{i \in S} w_t(i) y_i h_{t+1}(\mathbf{x}_i) \qquad \text{with } w_t(i) = \frac{c'(\rho(g_t(\mathbf{x}_i), y_i))}{\sum_{j \in S} c'(\rho(g_t(\mathbf{x}_j), y_j))} \qquad (3)$$

## 3 Generalizing MarginBoost to semi-supervised classification

### 3.1 Margin Extension

For labeled data, the margin measures the quality of the classifier output. When no label is observed, the usual margin cannot be calculated and has to be estimated. A first estimation could be derived from the expected margin $\mathbf{E}_y \rho_L(g_t(\mathbf{x}), y)$. We can use the output of the classifier $(g_t(\mathbf{x}) + 1)/2$ as an estimate of the posterior probability $P(Y = +1|\mathbf{x})$. This leads to the following margin $\rho_U^g$ which depends on the input and is linked with the response of the classifier :

Let $w_0(i) = 1/l$, $i = 1, \dots, l$.
Let $g_0(\mathbf{x}) = 0$
For $t = 1 \dots T$ (do the gradient descent):

1. Learn a gradient direction $h_{t+1} \in \mathcal{H}$ with a high value of
   $J_t^S = \sum_{i \in S} w_t(i) y_i h_{t+1}(\mathbf{x}_i)$
2. Apply the stopping rule : if $J_t^S \leq \sum_{i \in S} w_t(i) y_i g_t(\mathbf{x}_i)$ then return $g_t$ else go on.
3. Choose a step-length for the obtained direction by a line-search or by fixing it as a constant $\epsilon$
4. Add the new direction to obtain $g_{t+1} = \frac{(|\alpha_t| g_t + \alpha_{t+1} h_{t+1})}{|\alpha_{t+1}|}$
5. Fix the weight distribution : $w_{t+1} = \frac{c'(\rho(g_{t+1}(\mathbf{x}_i), y_i))}{\sum_{j \in S} c'(\rho(g_{t+1}(\mathbf{x}_j), y_j))}$

Figure 1: MarginBoost algorithm (with $L_1$ normalization of the combination coefficients)

$$\rho_U^g(g_t(\mathbf{x})) = \frac{(g_t(\mathbf{x}) + 1)}{2} g_t(\mathbf{x}) + (1 - \frac{(g_t(\mathbf{x}) + 1)}{2})(-g_t(\mathbf{x})) = g_t(\mathbf{x})^2 \qquad (4)$$

Another way of defining the extended margin is to use directly the maximum *a posteriori* estimate of the true margin. This MAP estimate depends on the sign of the classifier output and provides the following margin definition $\rho_U^s$ :

$$\rho_U^s(g_t(\mathbf{x})) = g_t(\mathbf{x}) \text{sign}(g_t(\mathbf{x})) = |g_t(\mathbf{x})| \qquad (5)$$

### 3.2 Semi-Supervised MarginBoost : generalization of marginBoost to deal with unlabeled data

The generalization of the margin can be used to define an appropriate cost functional for the semi-supervised learning task. Considering that the training sample $S$ is now divided into two disjoint subsets $L$ for labeled data and $U$ for unlabeled data, the cost falls into two parts involving $\rho_L = \rho$ and $\rho_U$:

$$C(g_t) = \sum_{i \in L} c(\rho_L(g_t(\mathbf{x}_i), y_i) + \sum_{i \in U} c(\rho_U(g_t(\mathbf{x}_i))) \qquad (6)$$

The maximization of $- < \nabla C(g_t), h_{t+1} >$ is equivalent to optimize the new quantity $J_t^S$ that falls now into two terms $J_t^S = J_t^L + J_t^U$. The first term one can be directly obtained from equation (3) :

$$J_t^L = \sum_{i \in L} w_t(i).y_i h_{t+1}(\mathbf{x}_i) \qquad \text{with } w_t(i) = \frac{c'(\rho(g_t(\mathbf{x}_i), y_i))}{\sum_{j \in L} c'(\rho(g_t(\mathbf{x}_j), y_j))} \qquad (7)$$

The second term, $J_t^U$, can be expressed as following:

$$J_t^U = \sum_{i \in U} w_t(i) \frac{\partial \rho_U(g_t(\mathbf{x}_i))}{\partial g_t(\mathbf{x}_i)} h_{t+1}(\mathbf{x}_i) \qquad (8)$$

with the weight distribution $w_t$ now defined as :

$$w_t(i) = \begin{cases} \frac{c'(\rho_L(g_t(\mathbf{x}_i),y_i))}{|w_t|} & \text{if } i \in L \\ \frac{c'(\rho_U(g_t(\mathbf{x}_i)))}{|w_t|} & \text{if } i \in U \end{cases} \quad \text{with } |w_t| = \sum_{i \in S} w_t(i) \qquad (9)$$

This expression of $J_t^S$ comes directly from differential calculus and the chosen inner product :

$$\nabla C(g_t)(\mathbf{x}_i) = \begin{cases} y_i c'(\rho_L(g_t(\mathbf{x}_i),y_i)) & \text{if } x = x_i \text{ and } i \in L \\ c'(\rho_U(g_t(\mathbf{x}_i)))\frac{\partial \rho_U(g_t(\mathbf{x}_i))}{\partial g_t(\mathbf{x}_i)} & \text{if } x = x_i \text{ and } i \in U \end{cases} \qquad (10)$$

Implementation of SSMBoost with margins $\rho_U^g$ and $\rho_U^s$ requires their derivatives. Let us notice that the "signed margin", $\rho_U{}^s$, is not derivable at point 0. However, according to the results of convex analysis (see for instance [2]), it is possible to define the "subderivative' of $\rho_U{}^s$ since it is a continuous and convex function. The value of the subderivative corresponds here to the average value of the right and left derivatives.

$$\frac{\partial \rho_U{}^s(g_t(\mathbf{x}_i))}{\partial g_t(\mathbf{x}_i)} = \begin{cases} \text{sign}(g(\mathbf{x}_i)) & \text{if } \mathbf{x} \neq 0 \\ 0 & \text{if } \mathbf{x} = 0 \end{cases} \qquad (11)$$

And, for the "squared margin" $\rho_U{}^g$, we have :

$$\frac{\partial \rho_U{}^g(g_t(\mathbf{x}_i))}{\partial g_t(\mathbf{x}_i)} = 2g(\mathbf{x}_i) \qquad (12)$$

This completes the set of ingredients that must be incorporated into the algorithm of Fig. 1 to obtain SSMBoost.

## 4 Base Classifier

The base classifier should be able to make use of the unlabeled data provided by the boosting algorithm. Mixture models are well suited for this purpose, as shown by their extensive use in clustering. Hierarchical mixtures provide flexible discrimination tools, where each conditional distribution $f(\mathbf{x}|y = k)$ is modelled by a mixture of components [4]. At the high level, the distribution is described by

$$f(\mathbf{x}; \Phi) = \sum_{k=1}^{K} p_k f_k(\mathbf{x}; \theta_k) \ , \qquad (13)$$

where $K$ is the number of classes, $p_k$ are the mixing proportions, $\theta_k$ the conditional distribution parameters, and $\Phi$ denotes all parameters $\{p_k; \theta_k\}_{k=1}^{K}$. The high-level description can also be expressed as a low-level mixture of components, as shown here for binary classification:

$$f(\mathbf{x}; \Phi) = \sum_{k_1=1}^{K_1} p_{k_1} f_{k_1}(\mathbf{x}; \theta_{k_1}) + \sum_{k_2=1}^{K_2} p_{k_2} f_{k_2}(\mathbf{x}; \theta_{k_2}) \ . \qquad (14)$$

With this setting, the EM algorithm is used to maximize the log-likelihood with respect to $\Phi$ considering the incomplete data is $\{\mathbf{x}_i, y_i\}_{i=1}^{l}$ and the missing data is the component label $c_{ik}$, $k = 1, \ldots, K_1 + K_2$ [11]. An original implementation of EM based on the concept of possible labels [1] is considered here. It is well adapted to hierarchical mixtures, where the class label $y$ provides a subset of possible components. When $y = 1$ the first $K_1$ modes are possible, when $y = -1$ the last $K_2$ modes are possible, and when an example is unlabeled, all modes are possible.

A binary vector $\mathbf{z}_i \in \{0, 1\}^{(K_1 + K_2)}$ indicates the components from which feature vector $\mathbf{x}_i$ may have been generated, in agreement with the assumed mixture model and the (absence of) label $y_i$. Assuming that the training sample $\{\mathbf{x}_i, \mathbf{z}_i\}_{i=1}^l$ is i.i.d, the weighted log-likelihood is given by

$$L(\Phi; \{\mathbf{x}_i, \mathbf{z}_i\}_{i=1}^l = \sum_{i=1}^l w_t(i) \log\left(f(\mathbf{x}_i, \mathbf{z}_i; \Phi)\right) \ , \tag{15}$$

where $w_t(i)$ are provided by boosting at step $t$. $L$ is maximized using the following EM algorithm:

**E-Step** Compute the expectation of $L(\Phi; \{\mathbf{x}_i, \mathbf{z}_i\}_{i=1}^l)$ conditionally to $\{\mathbf{x}_i, \mathbf{z}_i\}_{i=1}^l$ and the current value of $\Phi$ (denoted $\Phi^q$):

$$Q(\Phi|\Phi^q) = \sum_{i=1}^l \sum_{k=1}^{K_1+K_2} w_t(i) u_{ik} \log\left(p_k f_k(\mathbf{x}_i; \theta_k)\right) \ , \tag{16}$$

$$\text{with } u_{ik} = \frac{z_{ik} p_k f_k(\mathbf{x}_i; \theta_k)}{\sum_\ell z_{i\ell} p_\ell f_\ell(\mathbf{x}_i; \theta_\ell)} \ .$$

**M-Step** Maximize $Q(\Phi|\Phi^q)$ with respect to $\Phi$.

Assuming that each mode $k$ follows a normal distribution with mean $\boldsymbol{\mu}_k$, and covariance $\boldsymbol{\Sigma}_k$, $\Phi^{q+1} = \{\boldsymbol{\mu}_k^{q+1}; \boldsymbol{\Sigma}_k^{q+1}; p_k^{q+1}\}_{k=1}^{K_1+K_2}$ is given by:

$$p_k^{q+1} = \frac{\sum_i w_t(i) u_{ik}}{\sum_i w_t(i)} \ ; \ \boldsymbol{\mu}_k^{q+1} = \sum_i \frac{w_t(i) u_{ik} \mathbf{x}_i}{\sum_i w_t(i) u_{ik}} \ ; \tag{17}$$

$$\boldsymbol{\Sigma}_k^{q+1} = \frac{1}{\sum_i w_t(i) u_{ik}} \sum_{i=1}^l u_{ik}(\mathbf{x}_i - \boldsymbol{\mu}_k^{q+1})(\mathbf{x}_i - \boldsymbol{\mu}_k^{q+1})^t \ . \tag{18}$$

## 5 Experimental results

Tests of the algorithm are performed on three benchmarks of the boosting literature: twonorm and ringnorm [6] and banana [13]. Information about these datasets and the results obtained in discrimination are available at *www.first.gmd.de/~raetsch/*. 10 different samples were used for each experiment.

We first study the behavior of SSMBoost according the evolution of the test error with increasing rates of unlabeled data (table 1). We consider five different settings where 0%, 50%, 75%, 90% and 95% of labels are missing. SSMB is tested for the margins $\rho_U^g$ and $\rho_U^s$ with $c(\mathbf{x}) = exp(-\mathbf{x})$. It is compared to mixture models and AdaBoost. SSMBoost and AdaBoost are trained identically, the only difference being that AdaBoost is not provided with missing labels.

Both algorithms are run for $T = 100$ boosting steps, without special care of overfitting. The base classifier (called here base(EM)) is a hierarchical mixture model with an arbitrary choice of 4 modes per class but the algorithm (which may be stalled in local minima) is restarted 100 times from different initial solutions, and the best final solution (regarding training error rate) is selected. We report mean error rates together with the lower and upper quartiles in table 1. For sake of space, we did not display the results obtained without missing labels: in this case, AdaBoost and SSMBoost behave nearly identically and better than EM only for Banana.

For rates of unlabeled data inferior to 95%, SSMBoost beats slightly AdaBoost for Ringnorm and Twonorm (except for 75%) but is not able to do as well as

Table 1: Mean error rates (in %) and interquartiles obtained with 4 different percentages of unlabeled data for mixture models base(EM), AdaBoost and SSMBoost.

| Ringnorm | 50% | 75% | 90% | 95% |
|---|---|---|---|---|
| base(EM) | 2.1[ 1.7, 2.1] | 4.3[ 1.9, 5.7] | 9.5[ 2.7,12.0] | 23.7[14.5,27.0] |
| AdaBoost | 1.8[ 1.6, 2.0] | 3.1[ 1.9, 4.1] | 11.5[ 4.2,12.1] | 28.7[11.5,37.6] |
| SSMBoost $\rho^s$ | **1.7**[ 1.5, 1.8] | **2.0**[ 1.5, 2.4] | **3.7**[ 2.1, 4.8] | **6.9**[ 5.6,10.7] |
| SSMBoost $\rho^g$ | **1.7**[ 1.6, 1.8] | **2.0**[ 1.4, 2.5] | 4.5[ 2.2, 3.6] | 8.1[ 4.2, 9.0] |

| Twonorm | 50% | 75% | 90% | 95% |
|---|---|---|---|---|
| base(EM) | 3.2[ 2.7, 3.1] | 6.5[ 3.0, 9.0] | 20.6[10.3,22.5] | 24.8[18.3,31.9] |
| AdaBoost | 3.2[ 2.9, 3.2] | **3.2**[ 3.0, 3.5] | 11.0[ 5.2,14.2] | 38.9[29.4,50.0] |
| SSMBoost $\rho^s$ | **2.7**[ 2.5, 2.9] | 3.4[ 2.8, 4.3] | **10.1**[ 5.8,13.6] | **20.4**[11.9,32.3] |
| SSMBoost $\rho^g$ | **2.7**[ 2.5, 2.8] | 3.4[ 2.8, 4.2] | 11.0[ 5.6,16.2] | 21.1[12.5,30.8] |

| Banana | 50% | 75% | 90% | 95% |
|---|---|---|---|---|
| base(EM) | 18.2[16.7,18.6] | 21.8[18.0,25.0] | 26.1[20.7,29.8] | 31.7[23.8,35.8] |
| AdaBoost | **12.6**[11.7,13.1] | **15.2**[13.0,16.8] | **22.1**[18.0,24.3] | 37.5[32.2,42.2] |
| SSMBoost $\rho^s$ | 13.3[12.7,14.3] | 17.0[15.3,17.8] | 22.2[18.0,28.0] | **28.3**[20.2,35.2] |
| SSMBoost $\rho^g$ | 13.3[12.8,14.2] | 16.9[15.6,17.8] | 22.8[18.3,29.3] | 28.6[21.5,34.2] |

AdaBoost on Banana data. One possible explanation is that the discrimination frontiers involved in the banana problem are so complex that the labels really bring crucial informations and thus adding unlabeled data does not help in such a case.

Nevertheless, at rate 95% which is the most realistic situation, the margin $\rho_U^s$ obtains the minimal error rate for each of the three problems. It shows that it is worth boosting and using unlabeled data.

As there is no great difference between the two proposed margins, we conducted further experiments using only the $\rho_U^s$.

Second, in order to study the relation between the presence of noise in the dataset and the ability of SSMBoost to enhance generalization performance, we draw in Fig. 2, the test errors obtained for problems with different values of Bayes error when varying the rate of labeled examples. We see that even for difficult tasks (very noisy problems), the degradation in performance for large subsets of unlabeled data is still low. This reflects some consistency in the behavior of our algorithm.

Third, we test the sensibility of SSMBoost to overfitting. Overfitting can usually be avoided by techniques such as early stopping, softenizing of the margin ([13], [14]) or using an adequate margin function such as $1 - tanh(\rho)$ instead of $exp(-\rho)$ [10]. Here we keep using $c = exp$ and ran SSMBoost with a maximal number of step $T = 1000$ with 95% of unlabeled data. Of course, this does not correspond to a realistic use of boosting in practice but it allows to check if the algorithm behaves consistently in terms of gradient steps number. It is remarkable that no overfitting is observed and in the Twonorm case (see Fig. 3) , the test error still decreases ! We also observe that the standard error deviation is reduced at the end of the process. For the banana problem (see Fig. 3 b.), we observe a stabilization near the step $t = 100$. A massive presence of unlabeled data implies thus a regularizing effect.

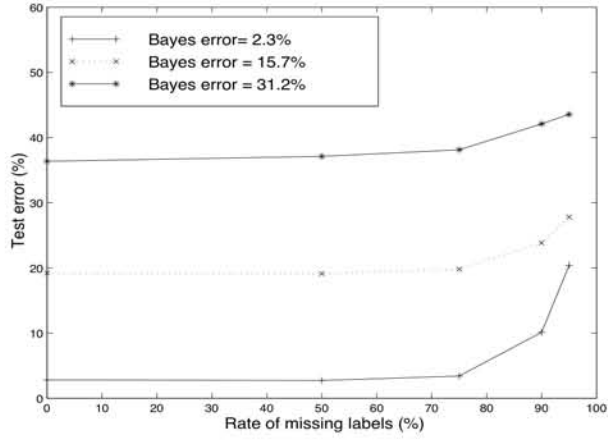

Figure 2: Consistency of the SSMBoost behavior: evolution of test error versus the missing labels rate with respect to various Bayes error (twonorm ).

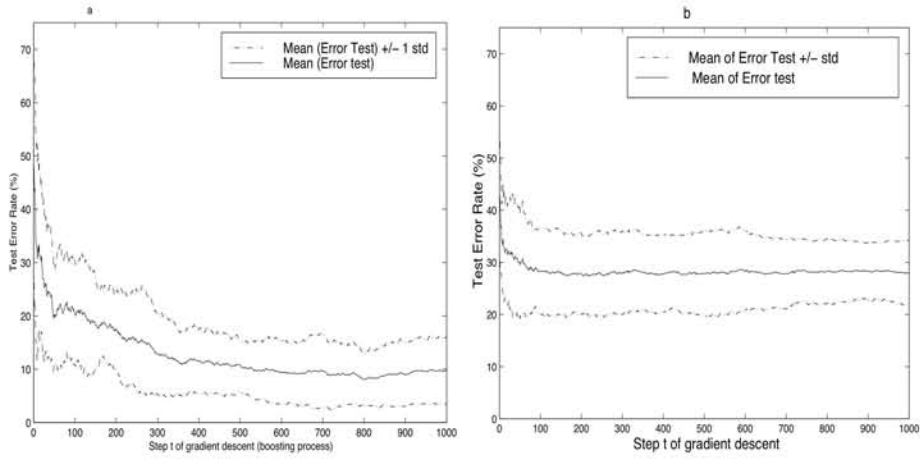

Figure 3: Evolution of Test error with respect to maximal number T of iterations with 95% of missing labels (Twonorm and Banana).

# 6 Conclusion

MarginBoost algorithm has been extended to deal with both labeled and unlabeled data. Results obtained on three classical benchmarks of boosting litterature show that it is worth using additional information conveyed by the patterns alone. No overfitting was observed during processing SSMBoost on the benchmarks when 95% of the labels are missing: this should mean that the unlabeled data should play a regularizing role in the ensemble classifier during the boosting process. After applying this method to a large real dataset such as those of text-categorization, our future works on this theme will concern the use of the extended margin cost function on the base classifiers itself like multilayered perceptrons or decision trees. Another approach could also be conducted from the more general framework of AnyBoost that optimize any differential cost function.

## Footnotes

[1] $\alpha_\tau > 0$ and $L_1$ norm is used for normalization: $|\alpha_\tau| = \sum_{\tau=1}^{t} \alpha_\tau$

[2] $< f, g >= \sum_{i \in S} f(\mathbf{x}_i) g(\mathbf{x}_i)$

# References

[1] C. Ambroise and G. Govaert. EM algorithm for partially known labels. In *IFCS 2000*, july 2000.

[2] J.-P. Aubin. *L'analyse non linéaire et ses applications à l'économie*. Masson, 1984.

[3] K. P. Bennett and A. Demiriz. Semi-supervised support vector machines. In D. Cohn, M. Kearns, and S. Solla, editors, *Advances in Neural Information Processing Systems*, pages 368–374. MIT Press, 1999.

[4] C.M. Bishop and M.E. Tipping. A hierarchical latent variable model for data vizualization. *IEEE PAMI*, 20:281–293, 1998.

[5] A. Blum and Tom Mitchell. Combining labeled and unlabeled data with co-training. In *Proceedings of the 1998 Conference on Computational Learning Theory*, July 1998.

[6] L. Breiman. Prediction games and arcing algorithms. Technical Report 504, Statistics Department, University of California at Berkeley, 1997.

[7] Y. Freund and R. E. Schapire. Experiments with a new boosting algorithm. In *Machine Learning: Proceedings of the Thirteenth International Conference*, pages 148–156. Morgan Kauffman, 1996.

[8] J. Friedman, T. Hastie, and R. Tibshirani. Additive logistic regression: a statistical view of boosting. *The Annals of Statistics*, 28(2):337–407, 2000.

[9] Y. Grandvalet, F. d'Alché Buc, and C. Ambroise. Boosting mixture models for semi-supervised learning. In *ICANN 2001*, august 2001.

[10] L. Mason, J. Baxter, P. L. Bartlett, and M. Frean. Functional gradient techniques for combining hypotheses. In *Advances in Large Margin Classifiers*. MIT, 2000.

[11] G.J. McLachlan and T. Krishnan. *The EM algorithm and extensions*. Wiley, 1997.

[12] K. Nigam, A. K. McCallum, S. Thrun, and T. Mitchell. Text classification from labeled and unlabeled documents using EM. *Machine learning*, 39(2/3):135–167, 2000.

[13] G. Rätsch, T. Onoda, and K.-R. Müller. Soft margins for AdaBoost. Technical report, Department of Computer Science, Royal Holloway, London, 1998.

[14] G. Rätsch, T. Onoda, and K.-R. Müller. Soft margins for AdaBoost. *Machine Learning*, 42(3):287–320, 2001.

[15] R. E. Schapire, Y. Freund, P. Bartlett, and W. S. Lee. Boosting the margin: A new explanation for the effectiveness of voting methods. *The Annals of Statistics*, 26(5):1651–1686, 1998.

[16] Matthias Seeger. Learning with labeled and unlabeled data, www.citeseer.nj.nec.com/seeger01learning.html.
